# Signal Estimation Under Random Time-Warpings and Nonlinear Signal Alignment

**Sebastian Kurtek**     **Anuj Srivastava**     **Wei Wu**
Department of Statistics
Florida State University, Tallahassee, FL 32306
skurtek,anuj,wwu@stat.fsu.edu

## Abstract

While signal estimation under random amplitudes, phase shifts, and additive noise is studied frequently, the problem of estimating a deterministic signal under random time-warpings has been relatively unexplored. We present a novel framework for estimating the unknown signal that utilizes the action of the warping group to form an equivalence relation between signals. First, we derive an estimator for the equivalence class of the unknown signal using the notion of Karcher mean on the quotient space of equivalence classes. This step requires the use of Fisher-Rao Riemannian metric and a square-root representation of signals to enable computations of distances and means under this metric. Then, we define a notion of the center of a class and show that the center of the estimated class is a consistent estimator of the underlying unknown signal. This estimation algorithm has many applications: (1) registration/alignment of functional data, (2) separation of phase/amplitude components of functional data, (3) joint demodulation and carrier estimation, and (4) sparse modeling of functional data. Here we demonstrate only (1) and (2): Given signals are temporally aligned using nonlinear warpings and, thus, separated into their phase and amplitude components. The proposed method for signal alignment is shown to have state of the art performance using Berkeley growth, handwritten signatures, and neuroscience spike train data.

## 1  Introduction

Consider the problem of estimating signal using noisy observation under the model:

$$f(t) = cg(a\,t - \phi) + e(t)\,,$$

where the random quantities $c \in \mathbb{R}$ is the scale, $a \in \mathbb{R}$ is the rate, $\phi \in \mathbb{R}$ is the phase shift, and $e(t) \in \mathbb{R}$ is the additive noise. There has been an elaborate theory for estimation of the underlying signal $g$, given one or several observations of the function $f$. Often one assumes that $g$ takes a parametric form, e.g. a superposition of Gaussians or exponentials with different parameters, and estimates these parameters from the observed data [12]. For instance, the estimation of sinusoids or exponentials in additive Gaussian noise is a classical problem in signal and speech processing. In this paper we consider a related but fundamentally different estimation problem where the observed functional data is modeled as: for $t \in [0, 1]$,

$$f_i(t) = c_i g(\gamma_i(t)) + e_i, \quad i = 1, 2, \ldots, n\,, \tag{1}$$

Here $\gamma_i : [0, 1] \to [0, 1]$ are diffeomorphisms with $\gamma_i(0) = 0$ and $\gamma_i(1) = 1$. The $f_i$s represent observations of an unknown, deterministic signal $g$ under random warpings $\gamma_i$, scalings $c_i$ and vertical translations $e_i \in \mathbb{R}$. (A more general model would be to use full functions for additive noise but that requires further discussion due to identifiability issues. Thus, we restrict to the above model in this paper.) This problem is interesting because in many situations, including speech, SONAR, RADAR,

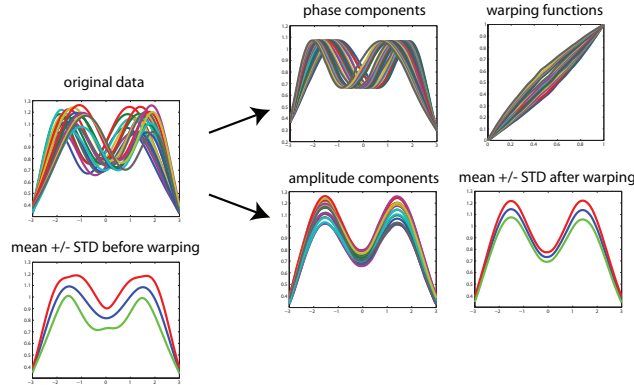

Figure 1: Separation of phase and amplitude variability in function data.

NMR, fMRI, and MEG applications, the noise can actually affect the instantaneous phase of the signal, resulting in an observation that is a phase (or frequency) modulation of the original signal. This problem is challenging because of the nonparametric, random nature of the warping functions $\gamma_i$s. It seems difficult to be able to recover $g$ where its observations have been time-warped nonlinearly in a random fashion. The past papers have either restricted to linear warpings (e.g. $\gamma_i(t) = a_i t - \phi_i$) or known $g$ (e.g. $g(t) = cos(t)$). It turns out that without any further restrictions on $\gamma_i$s one can recover $g$ only up to an arbitrary warping function. This is easy to see since $g \circ \gamma_i = (g \circ \gamma) \circ (\gamma^{-1} \circ \gamma_i)$ for any warping function $\gamma$. (As described later, the warping functions are restricted to be automorphisms of a domain and, hence, form a group.) Under an additional condition related to the mean of (inverses of) $\gamma_i$s, we can reach the exact signal $g$, as demonstrated in this paper.

In fact, this model describes several related, some even equivalent, problems but with distinct applications:

**Problem 1: Joint Phase Demodulation and Carrier Estimation:** One can view this problem as that of *phase (or frequency) demodulation* but without the knowledge of the carrier signal $g$. Thus, it becomes a problem of joint estimation of the carrier signal ($g$) and phase demodulation ($\gamma_i^{-1}$) of signals that share the same carrier. In case the carrier signal $g$ is known, e.g. $g$ is a sinusoid, then it is relatively easier to estimate the warping functions using dynamic time warping or other estimation theoretic methods [15, 13]. So, we consider problem of estimation of $g$ from $\{f_i\}$ under the model given in Eqn. 1.

**Problem 2: Phase-Amplitude Separation**: Consider the set of signals shown in the top-left panel of Fig. 1. These functions differ from each other in both heights and locations of their peaks and valleys. One would like to separate the variability associated with the heights, called the *amplitude* variability, from the variability associated with the locations, termed the *phase* variability. Although this problem has been studied for almost two decades in the statistics community, see e.g. [7, 9, 4, 11, 8], it is still considered an open problem. Extracting the amplitude variability implies temporally aligning the given functions using nonlinear time warping, with the result shown in the bottom right. The corresponding set of warping functions, shown in the top right, represent the phase variability. The phase component can also be illustrated by applying these warping functions to the same function, also shown in the top right. The main reason for separating functional data into these components is to better preserve the structure of the observed data, since a separate modeling of amplitude and phase variability will be more natural, parsimonious and efficient. It may not be obvious but the solution to this separation problem is intimately connected to the estimation of $g$ in Eqn. 1.

**Problem 3: Multiple Signal/Image Registration**: The problem of phase-amplitude separation is intrinsically same as the problem of joint registration of multiple signals. The problem here is: Given a set of observed signals $\{f_i\}$ estimate the corresponding points in their domains. In other words,

what are the $\gamma_i$s such that, for any $t_0$, $f_i(\gamma_i^{-1}(t_0))$ correspond to each other. The bottom right panels of Fig. 1 show the registered signals. Although this problem is more commonly studied for images, its one-dimensional version is non-trivial and helps understand the basic challenges. We will study the 1D problem in this paper but, at least conceptually, the solutions extend to higher-dimensional problems also.

In this paper we provide the following specific contributions. We study the problem of estimating $g$ given a set $\{f_i\}$ under the model given in Eqn. 1 and propose a consistent estimator for this problem, along with the supporting asymptotic theory. Also, we illustrate the use of this solution in automated alignment of sets of given signals. Our framework is based on an equivalence relation between signals defined as follows. Two signals, are deemed equivalent if one can be time-warped into the other; since the warping functions form a group, the equivalence class is an orbit of the warping group. This relation partitions the set of signals into equivalence classes, and the set of equivalence classes (orbits) forms a quotient space. Our estimation of $g$ is based on two steps. First, we estimate the equivalence class of $g$ using the notion of Karcher mean on quotient space which, in turn, requires a distance on this quotient space. This distance should respect the equivalence structure, i.e. the distance between any elements should be zero if and only if they are in the same class. We propose to use a distance that results from the Fisher-Rao Riemannian metric. This metric was introduced in 1945 by C. R. Rao [10] and studied rigorously in the 70s and 80s by Amari [1], Efron [3], Kass [6], Cencov [2], and others. While those earlier efforts were focused on analyzing parametric families, we use the *nonparametric* version of the Fisher-Rao Riemannian metric in this paper. The difficulty in using this metric directly is that it is not straightforward to compute geodesics (remember that geodesics lengths provide the desired distances). However, a simple square-root transformation converts this metric into the standard $\mathbb{L}^2$ metric and the distance is obtainable as a simple $\mathbb{L}^2$ norm between the square-root forms of functions. Second, given an estimate of the equivalence class of $g$, we define the notion of a center of an orbit and use that to derive an estimator for $g$.

## 2 Background Material

We introduce some notation. Let $\Gamma$ be the set of orientation-preserving diffeomorphisms of the unit interval $[0,1]$: $\Gamma = \{\gamma : [0,1] \to [0,1] | \gamma(0) = 0, \ \gamma(1) = 1, \ \gamma \text{ is a diffeo}\}$. Elements of $\Gamma$ form a group, i.e. (1) for any $\gamma_1, \gamma_2 \in \Gamma$, their composition $\gamma_1 \circ \gamma_2 \in \Gamma$; and (2) for any $\gamma \in \Gamma$, its inverse $\gamma^{-1} \in \Gamma$, where the identity is the self-mapping $\gamma_{id}(t) = t$. We will use $\|f\|$ to denote the $\mathbb{L}^2$ norm $(\int_0^1 |f(t)|^2 dt)^{1/2}$.

### 2.1 Representation Space of Functions

Let $f$ be a real-valued function on the interval $[0,1]$. We are going to restrict to those $f$ that are absolutely continuous on $[0,1]$; let $\mathcal{F}$ denote the set of all such functions. We define a mapping: $Q : \mathbb{R} \to \mathbb{R}$ according to: $Q(x) \equiv \begin{cases} x/\sqrt{|x|} & \text{if } |x| \neq 0 \\ 0 & \text{otherwise} \end{cases}$. Note that $Q$ is a continuous map. For the purpose of studying the function $f$, we will represent it using a square-root velocity function (SRVF) defined as $q : [0,1] \to \mathbb{R}$, where $q(t) \equiv Q(\dot{f}(t)) = \dot{f}(t)/\sqrt{|\dot{f}(t)|}$. It can be shown that if the function $f$ is absolutely continuous, then the resulting SRVF is square integrable. Thus, we will define $\mathbb{L}^2([0,1], \mathbb{R})$ (or simply $\mathbb{L}^2$) to be the set of all SRVFs. For every $q \in \mathbb{L}^2$ there exists a function $f$ (unique up to a constant, or a vertical translation) such that the given $q$ is the SRVF of that $f$. If we warp a function $f$ by $\gamma$, the SRVF of $f \circ \gamma$ is given by: $\tilde{q}(t) = \frac{\frac{d}{dt}(f \circ \gamma)(t)}{\sqrt{|\frac{d}{dt}(f \circ \gamma)(t)|}} = (q \circ \gamma)(t)\sqrt{\dot{\gamma}(t)}$. We will denote this transformation by $(q, \gamma) = (q \circ \gamma)\sqrt{\dot{\gamma}}$.

### 2.2 Elastic Riemannian Metric

**Definition 1** *For any $f \in \mathcal{F}$ and $v_1, v_2 \in T_f(\mathcal{F})$, where $T_f(\mathcal{F})$ is the tangent space to $\mathcal{F}$ at $f$, the Fisher-Rao Riemannian metric is defined as the inner product:*

$$\langle\langle v_1, v_2 \rangle\rangle_f = \frac{1}{4} \int_0^1 \dot{v}_1(t)\dot{v}_2(t)\frac{1}{|\dot{f}(t)|}dt \ . \tag{2}$$

This metric has many fundamental advantages, including the fact that it is the only Riemannian metric that is invariant to the domain warping [2]. This metric is somewhat complicated since it changes from point to point on $\mathcal{F}$, and it is not straightforward to derive equations for computing geodesics in $\mathcal{F}$. However, a small transformation provide an enormous simplification of this task. This motivates the use of SRVFs for representing and aligning elastic functions.

**Lemma 1** *Under the SRVF representation, the Fisher-Rao Riemannian metric becomes the standard $\mathbb{L}^2$ metric.*

This result can be used to compute the distance $d_{FR}$ between any two functions by computing the $\mathbb{L}^2$ distance between the corresponding SRVFs, that is, $d_{FR}(f_1, f_2) = \|q_1 - q_2\|$. The next question is: What is the effect of warping on $d_{FR}$? This is answered by the following result of *isometry*.

**Lemma 2** *For any two SRVFs $q_1, q_2 \in \mathbb{L}^2$ and $\gamma \in \Gamma$, $\|(q_1, \gamma) - (q_2, \gamma)\| = \|q_1 - q_2\|$.*

### 2.3   Elastic Distance on Quotient Space

Our next step is to define an elastic distance between functions as follows. The orbit of an SRVF $q \in \mathbb{L}^2$ is given by: $[q] = \text{closure}\{(q, \gamma)|\gamma \in \Gamma\}$. It is the set of SRVFs associated with all the warpings of a function, and their limit points. Let $\mathcal{S}$ denote the set of all such orbits. To compare any two orbits we need a metric on $\mathcal{S}$. We will use the Fisher-Rao distance to induce a distance between orbits, and we can do that only because under this the action of $\Gamma$ is by isometries.

**Definition 2** *For any two functions $f_1$, $f_2 \in \mathcal{F}$ and the corresponding SRVFs, $q_1, q_2 \in \mathbb{L}^2$, we define the elastic distance $d$ on the quotient space $\mathcal{S}$ to be: $d([q_1], [q_2]) = \inf_{\gamma \in \Gamma} \|q_1 - (q_2, \gamma)\|$.*

Note that the distance $d$ between a function and its domain-warped version is zero. However, it can be shown that if two SRVFs belong to different orbits, then the distance between them is non-zero. Thus, this distance $d$ is a proper distance (i.e. it satisfies non-negativity, symmetry, and the triangle inequality) on $\mathcal{S}$ but not on $\mathbb{L}^2$ itself, where it is only a pseudo-distance.

## 3   Signal Estimation Method

Our estimation is based on the model $f_i = c_i(g \circ \gamma_i) + e_i$, $i = 1, \cdots, n$, where $g, f_i \in \mathcal{F}$, $c_i \in \mathbb{R}_+$, $\gamma_i \in \Gamma$ and $e_i \in \mathbb{R}$. Given $\{f_i\}$, our goal is to identify warping functions $\{\gamma_i\}$ so as to reconstruct $g$. We will do so in **three steps**: 1) For a given collection of functions $\{f_i\}$, and their SRVFs $\{q_i\}$, we compute the mean of the corresponding orbits $\{[q_i]\}$ in the quotient space $\mathcal{S}$; we will call it $[\mu]_n$. 2) We compute an appropriate element of this mean orbit to define a template $\mu_n$ in $\mathbb{L}^2$. The optimal warping functions $\{\gamma_i\}$ are estimated by align individual functions to match the template $\mu_n$. 3) The estimated warping functions are then used to align $\{f_i\}$ and reconstruct the underlying signal $g$.

### 3.1   Pre-step: Karcher Mean of Points in $\Gamma$

In this section we will define a Karcher mean of a set of warping functions $\{\gamma_i\}$, under the Fisher-Rao metric, using the differential geometry of $\Gamma$. Analysis on $\Gamma$ is not straightforward because it is a nonlinear manifold. To understand its geometry, we will represent an element $\gamma \in \Gamma$ by the square-root of its derivative $\psi = \sqrt{\dot{\gamma}}$. Note that this is the same as the SRVF defined earlier for elements of $\mathcal{F}$, except that $\dot{\gamma} > 0$ here. Since $\gamma(0) = 0$, the mapping from $\gamma$ to $\psi$ is a bijection and one can reconstruct $\gamma$ from $\psi$ using $\gamma(t) = \int_0^t \psi(s)^2 ds$. An important advantage of this transformation is that since $\|\psi\|^2 = \int_0^1 \psi(t)^2 dt = \int_0^1 \dot{\gamma}(t) dt = \gamma(1) - \gamma(0) = 1$, the set of all such $\psi$s is $\mathbb{S}_\infty$, the unit sphere in the Hilbert space $\mathbb{L}^2$. In other words, the square-root representation simplifies the complicated geometry of $\Gamma$ to the unit sphere. Recall that the distance between any two points on the unit sphere, under the Euclidean metric, is simply the length of the shortest arc of a great circle connecting them on the sphere. Using Lemma 1, the Fisher-Rao distance between any two warping functions is found to be $d_{FR}(\gamma_1, \gamma_2) = \cos^{-1}(\int_0^1 \sqrt{\dot{\gamma}_1(t)} \sqrt{\dot{\gamma}_2(t)} dt)$. Now that we have a proper distance on $\Gamma$, we can define a Karcher mean as follows.

**Definition 3** *For a given set of warping functions $\gamma_1, \gamma_2, \ldots, \gamma_n \in \Gamma$, define their Karcher mean to be $\bar{\gamma}_n = \text{argmin}_{\gamma \in \Gamma} \sum_{i=1}^n d_{FR}(\gamma, \gamma_i)^2$.*

The search for this minimum is performed using a standard iterative algorithm that is not repeated here to save space.

## 3.2 Step 1: Karcher Mean of Points in $\mathcal{S} = \mathbb{L}^2/\Gamma$

Next we consider the problem of finding means of points in the quotient space $\mathcal{S}$.

**Definition 4** *Define the Karcher mean $[\mu]_n$ of the given SRVF orbits $\{[q_i]\}$ in the space $\mathcal{S}$ as a local minimum of the sum of squares of elastic distances:*

$$[\mu]_n = \operatorname*{argmin}_{[q] \in \mathcal{S}} \sum_{i=1}^{n} d([q], [q_i])^2 . \tag{3}$$

We emphasize that the Karcher mean $[\mu]_n$ is actually an orbit of functions, rather than a function. The full algorithm for computing the Karcher mean in $\mathcal{S}$ is given next.

**Algorithm 1: Karcher Mean of $\{[q_i]\}$ in $\mathcal{S}$**

1. Initialization Step: Select $\mu = q_j$, where $j$ is any index in $\operatorname{argmin}_{1 \le i \le n} ||q_i - \frac{1}{n} \sum_{k=1}^{n} q_k||$.

2. For each $q_i$ find $\gamma_i^*$ by solving: $\gamma_i^* = \operatorname{argmin}_{\gamma \in \Gamma} ||\mu - (q_i, \gamma)||$. The solution to this optimization comes from a dynamic programming algorithm in a discretized domain.

3. Compute the aligned SRVFs using $\tilde{q}_i \mapsto (q_i, \gamma_i^*)$.

4. If the increment $||\frac{1}{n} \sum_{i=1}^{n} \tilde{q}_i - \mu||$ is small, then stop. Else, update the mean using $\mu \mapsto \frac{1}{n} \sum_{i=1}^{n} \tilde{q}_i$ and return to step 2.

The iterative update in Steps 2-4 is based on the gradient of the cost function given in Eqn. 3. Denote the estimated mean in the $k$th iteration by $\mu^{(k)}$. In the $k$th iteration, let $\gamma_i^{(k)}$ denote the optimal domain warping from $q_i$ to $\mu^{(k)}$ and let $\tilde{q}_i^{(k)} = (q_i, \gamma_i^{(k)})$. Then, $\sum_{i=1}^{n} d([\mu^{(k)}], [q_i])^2 = \sum_{i=1}^{n} ||\mu^{(k)} - \tilde{q}_i^{(k)}||^2 \ge \sum_{i=1}^{n} ||\mu^{(k+1)} - \tilde{q}_i^{(k)}||^2 \ge \sum_{i=1}^{n} d([\mu^{(k+1)}], [q_i])^2$. Thus, the cost function decreases iteratively and as zero is a lower bound, $\sum_{i=1}^{n} d([\mu^{(k)}], [q_i])^2$ will always converge.

## 3.3 Step 2: Center of an Orbit

Here we find a particular element of this mean orbit so that it can be used as a template to align the given functions.

**Definition 5** *For a given set of SRVFs $q_1, q_2, \ldots, q_n$ and $q$, define an element $\tilde{q}$ of $[q]$ as the center of $[q]$ with respect to the set $\{q_i\}$ if the warping functions $\{\gamma_i\}$, where $\gamma_i = \operatorname{argmin}_{\gamma \in \Gamma} ||\tilde{q} - (q_i, \gamma)||$, have the Karcher mean $\gamma_{id}$.*

We will prove the existence of such an element by construction.

**Algorithm 2: Finding Center of an Orbit** : WLOG, let $q$ be any element of the orbit $[q]$.

1. For each $q_i$ find $\gamma_i$ by solving: $\gamma_i = \operatorname{argmin}_{\gamma \in \Gamma} ||q - (q_i, \gamma)||$.

2. Compute the mean $\bar{\gamma}_n$ of all $\{\gamma_i\}$. The center of $[q]$ wrt $\{q_i\}$ is given by $\tilde{q} = (q, \bar{\gamma}_n^{-1})$.

We need to show that $\tilde{q}$ resulting from Algorithm 2 satisfies the mean condition in Definition 5. Note that $\gamma_i$ is chosen to minimize $||q - (q_i, \gamma)||$, and also that $||\tilde{q} - (q_i, \gamma)|| = ||(q, \bar{\gamma}_n^{-1}) - (q_i, \gamma)|| = ||q - (q_i, \gamma \circ \bar{\gamma}_n)||$. Therefore, $\gamma_i^* = \gamma_i \circ \bar{\gamma}_n^{-1}$ minimizes $||\tilde{q} - (q_i, \gamma)||$. That is, $\gamma_i^*$ is a warping that aligns $q_i$ to $\tilde{q}$. To verify the Karcher mean of $\gamma_i^*$, we compute the sum of squared distances $\sum_{i=1}^{n} d_{FR}(\gamma, \gamma_i^*)^2 = \sum_{i=1}^{n} d_{FR}(\gamma, \gamma_i \circ \bar{\gamma}_n^{-1})^2 = \sum_{i=1}^{n} d_{FR}(\gamma \circ \bar{\gamma}_n, \gamma_i)^2$. As $\bar{\gamma}_n$ is already the mean of $\gamma_i$, this sum of squares is minimized when $\gamma = \gamma_{id}$. That is, the mean of $\gamma_i^*$ is $\gamma_{id}$.

We will apply this setup in our problem by finding the center of $[\mu]_n$ with respect to the SRVFs $\{q_i\}$.

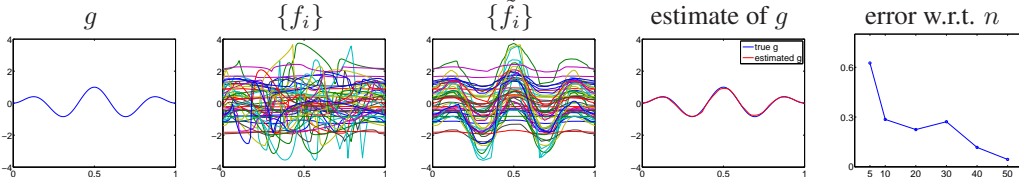

Figure 2: Example of consistent estimation.

## 3.4   Steps 1-3: Complete Estimation Algorithm

Consider the observation model $f_i = c_i(g \circ \gamma_i) + e_i$, $i = 1, \ldots, n$, where $g$ is an unknown signal, and $c_i \in \mathbb{R}_+, \gamma_i \in \Gamma$ and $e_i \in \mathbb{R}$ are random. Given the observations $\{f_i\}$, the goal is to estimate the signal $g$. To make the system identifiable, we need some constraints on $\gamma_i$, $c_i$, and $e_i$. In this paper, the constraints are: 1) the population mean of $\{\gamma_i^{-1}\}$ is identity $\gamma_{id}$, and 2) the population Karcher means of $\{c_i\}$ and $\{e_i\}$ are known, denoted by $E(\bar{c})$ and $E(\bar{e})$, respectively. Now we can utilize Algorithms 1 and 2 to present the full procedure for function alignment and signal estimation.

**Complete Estimation Algorithm**: Given a set of functions $\{f_i\}_{i=1}^n$ on $[0, 1]$, and population means $E(\bar{c})$ and $E(\bar{e})$. Let $\{q_i\}_{i=1}^n$ denote the SRVFs of $\{f_i\}_{i=1}^n$, respectively.

1. Computer the Karcher mean of $\{[q_i]\}$ in $\mathcal{S}$ using Algorithm 1; Denote it by $[\mu]_n$.

2. Find the center of $[\mu]_n$ *wrt* $\{q_i\}$ using Algorithm 2; call it $\mu_n$.

3. For $i = 1, 2, \ldots, n$, find $\gamma_i^*$ by solving: $\gamma_i^* = \operatorname{argmin}_{\gamma \in \Gamma} \|\mu_n - (q_i, \gamma)\|$.

4. Compute the aligned SRVFs $\tilde{q}_i = (q_i, \gamma_i^*)$ and aligned functions $\tilde{f}_i = f_i \circ \gamma_i^*$.

5. Return the warping functions $\{\gamma_i^*\}$ and the estimated signal $\hat{g} = (\frac{1}{n} \sum_{i=1}^n \tilde{f}_i - E(\bar{e}))/E(\bar{c})$.

**Illustration.**   We illustrate the estimation process using an example which is a quadratically-enveloped sine-wave function $g(t) = (1 - (1 - 2t)^2) \sin(5\pi t)$, $t \in [0, 1]$. We randomly generate $n = 50$ warping functions $\{\gamma_i\}$ such that $\{\gamma_i^{-1}\}$ are *i.i.d* with mean $\gamma_{id}$. We also generate *i.i.d* sequences $\{c_i\}$ and $\{e_i\}$ from the exponential distribution with mean 1 and the standard normal distribution, respectively. Then we compute functions $f_i = c_i(g \circ \gamma_i) + e_i$ to form the functional data. In Fig. 2, the first panel shows the function $g$, and the second panel shows the data $\{f_i\}$. The Complete Estimation Algorithm results in the aligned functions $\{\tilde{f}_i = f_i \circ \gamma_i^*\}$ that are are shown in the third panel in Fig. 2. In this case, $E(\bar{c})) = 1, E(\bar{e}) = 0$. This estimated $g$ (red) using the Complete Estimation Algorithm as well as the true $g$ (blue) are shown in the fourth panel. Note that the estimate is very successful despite large variability in the raw data. Finally, we examine the performance of the estimator with respect to the sample size, by performing this estimation for $n$ equal to $5, 10, 20, 30$, and $40$. The estimation errors, computed using the $\mathbb{L}^2$ norm between estimated $g$'s and the true $g$, are shown in the last panel. As we will show in the following theoretical development, this estimate converges to the true $g$ when the sample size $n$ grows large.

## 4   Estimator Consistency and Asymptotics

In this section we mathematically demonstrate that the proposed algorithms in Section 3 provide a consistent estimator for the underlying function $g$. This or related problems have been considered previously by several papers, including [14, 9], but we are not aware of any formal statistical solution.

At first, we establish the following useful result.

**Lemma 3** *For any $q_1, q_2 \in \mathbb{L}^2$ and a constant $c > 0$, we have $\operatorname{argmin}_{\gamma \in \Gamma} \|q_1 - (q_2, \gamma)\| = \operatorname{argmin}_{\gamma \in \Gamma} \|cq_1 - (q_2, \gamma)\|$.*

**Corollary 1** *For any function $q \in \mathbb{L}^2$ and constant $c > 0$, we have $\gamma_{id} \in \operatorname{argmin}_{\gamma \in \Gamma} \|cq - (q, \gamma)\|$. Moreover, if the set $\{t \in [0, 1] | q(t) = 0\}$ has (Lebesgue) measure 0, $\gamma_{id} = \operatorname{argmin}_{\gamma \in \Gamma} \|cq - (q, \gamma)\|$.*

Based on Lemma 3 and Corollary 1, we have the following result on the Karcher mean in the quotient space $\mathcal{S}$.

**Theorem 1** *For a function $g$, consider a sequence of functions $f_i(t) = c_i g(\gamma_i(t)) + e_i$, where $c_i$ is a positive constant, $e_i$ is a constant, and $\gamma_i$ is a time warping, $i = 1, \cdots, n$. Denote by $q_g$ and $q_i$ the SRVFs of $g$ and $f_i$, respectively, and let $\bar{s} = \frac{1}{n} \sum_{i=1}^{n} \sqrt{c_i}$. Then, the Karcher mean of $\{[q_i], i = 1, 2, \ldots, n\}$ in $\mathcal{S}$ is $\bar{s}[q_g]$. That is,*

$$[\mu]_n \equiv \underset{[q]}{\operatorname{argmin}} \left( \sum_{i=1}^{N} d^2([q_i], [q]) \right) = \bar{s}[q_g] = \bar{s}\{(q_g, \gamma), \gamma \in \Gamma\} .$$

Next, we present a simple fact about the Karcher mean (see Definition 3) of warping functions.

**Lemma 4** *Given a set $\{\gamma_i \in \Gamma | i = 1, ..., n\}$ and a $\gamma_0 \in \Gamma$, if the Karcher mean of $\{\gamma_i\}$ is $\bar{\gamma}$, then the Karcher mean of $\{\gamma_i \circ \gamma_0\}$ is $\bar{\gamma} \circ \gamma_0$.*

Theorem 1 ensures that $[\mu]_n$ belongs to the orbit of $[q_g]$ (up to a scale factor) but we are interested in estimating $g$ itself, rather than its orbit. We will show in two steps (Theorems 2 and 3) that finding the center of the orbit $[\mu]_n$ leads to a consistent estimator for $g$.

**Theorem 2** *Under the same conditions as in Theorem 1, let $\mu = (\bar{s}q_g, \gamma_0)$, for $\gamma_0 \in \Gamma$, denote an arbitrary element of the Karcher mean class $[\mu]_n = \bar{s}[q_g]$. Assume that the set $\{t \in [0,1] | \dot{g}(t) = 0\}$ has Lebesgue measure zero. If the population Karcher mean of $\{\gamma_i^{-1}\}$ is $\gamma_{id}$, then the center of the orbit $[\mu]_n$, denoted by $\mu_n$, satisfies $\lim_{n \to \infty} \mu_n = E(\bar{s})q_g$.*

This result shows that asymptotically one can recover the SRVF of the original signal by the Karcher mean of the SRVFs of the observed signals. Next in Theorem 3, we will show that one can also reconstruct $g$ using aligned functions $\{\tilde{f}_i\}$ generated by the Alignment Algorithm in Section 3.

**Theorem 3** *Under the same conditions as in Theorem 2, let $\gamma_i^* = \operatorname{argmin}_\gamma \|(q_i, \gamma) - \mu_n\|$ and $\tilde{f}_i = f_i \circ \gamma_i^*$. If we denote $\bar{c} = \frac{1}{n} \sum_{i=1}^{n} c_i$ and $\bar{e} = \frac{1}{n} \sum_{i=1}^{n} e_i$, then $\lim_{n \to \infty} \frac{1}{n} \sum_{i=1}^{n} \tilde{f}_i = E(\bar{c})g + E(\bar{e})$.*

## 5 Application to Signal Alignment

In this section we will focus on function alignment and comparison of alignment performance with some previous methods on several datasets. In this case, the given signals are viewed as $\{f_i\}$ in the previous set up and we estimate the center of the orbit and then use it for alignment of all signals. The datasets include 3 real experimental applications listed below. The data are shown in Column 1 in Fig. 3.

1. **Real Data 1. Berkeley Growth Data**: The Berkeley growth dataset for 39 male subjects [11]. For better illustrations, we have used the first derivatives of the growth (i.e. growth velocity) curves as the functions $\{f_i\}$ in our analysis.

2. **Real Data 2. Handwriting Signature Data**: 20 handwritten signatures and the acceleration functions along the signature curves [8]. Let $(x(t), y(t))$ denote the $x$ and $y$ coordinates of a signature traced as a function of time $t$. We study the acceleration functions $f(t) = \sqrt{\ddot{x}(t)^2 + \ddot{y}(t)^2}$ of the signatures.

3. **Real Data 3. Neural Spike Data**: Spiking activity of one motor cortical neuron in a Macaque monkey was recorded during arm-movement behavior [16]. The smoothed (using a Gaussian kernel) spike trains over 10 movement trials are used in this alignment analysis.

There are no standard criteria on evaluating function alignment in the current literature. Here we use the following three criteria so that together they provide a comprehensive evaluation, where $f_i$ and $\tilde{f}_i, i = 1, ..., N$, denote the original and the aligned functions, respectively.

1. **Least Squares**: $ls = \frac{1}{N} \sum_{i=1}^{N} \frac{\int (\tilde{f}_i(t) - \frac{1}{N-1} \sum_{j \neq i} \tilde{f}_j(t))^2 dt}{\int (f_i(t) - \frac{1}{N-1} \sum_{j \neq i} f_j(t))^2 dt}$. $ls$ measures the cross-sectional variance of the aligned functions, relative to original values. The smaller the value of $ls$, the better the alignment is in general.

| Original | PACE [11] | SMR [4] | MBM [5] | F-R |
|---|---|---|---|---|
| 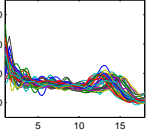 | 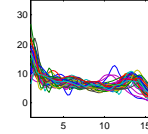 | 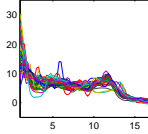 | 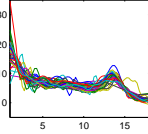 | 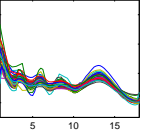 |
| Growth-male | (0.91, 1.09, 0.68) | (**0.45**, 1.17, 0.77) | (0.70, 1.17, 0.62) | (0.64, **1.18, 0.31**) |
| 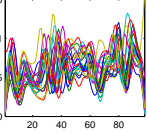 | 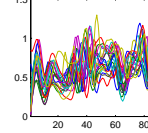 | 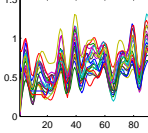 | 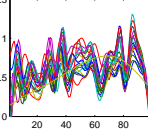 | 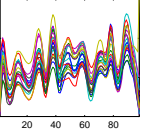 |
| Signature | (0.91, 1.18, 0.84) | (0.62, 1.59, **0.31**) | (0.64, 1.57, 0.46) | (**0.56, 1.79, 0.31**) |
| 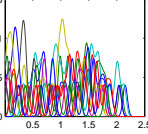 | 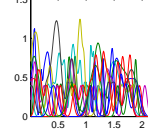 | 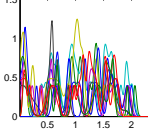 | 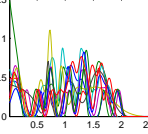 | 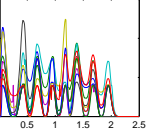 |
| Neural data | (0.87, 1.35, 1.10) | (0.69, 2.54, 0.95) | (0.48, 3.06, 0.40) | (**0.40, 3.77, 0.28**) |

Figure 3: Empirical evaluation of four methods on 3 real datasets, with the alignment performance computed using three criteria $(ls, pc, sls)$. The best cases are shown in boldface.

2. **Pairwise Correlation**: $pc = \frac{\sum_{i \neq j} cc(\tilde{f}_i(t), \tilde{f}_j(t))}{\sum_{i \neq j} cc(f_i(t), f_j(t))}$ , where $cc(f, g)$ is the pairwise Pearson's correlation between functions. Large values of $pc$ indicate good sychronization.

3. **Sobolev Least Squares**: $sls = \frac{\sum_{i=1}^{N} \int (\dot{\tilde{f}}_i(t) - \frac{1}{N} \sum_{j=1}^{N} \dot{\tilde{f}}_j)^2 dt}{\sum_{i=1}^{N} \int (\dot{f}_i(t) - \frac{1}{N} \sum_{j=1}^{N} \dot{f}_j)^2 dt}$ , This criterion measures the total cross-sectional variance of the derivatives of the aligned functions, relative to the original value. The smaller the value of $sls$, the better synchronization the method achieves.

We compare our Fisher-Rao (F-R) method with the Tang-Müller method [11] provided in principal analysis by conditional expectation (PACE) package, the self-modeling registration (SMR) method presented in [4], and the moment-based matching (MBM) technique presented in [5]. Fig. 3 summarizes the values of $(ls, pc, sls)$ for these four methods using 3 real datasets. From the results, we can see that the F-R method does uniformly well in functional alignment under all the evaluation metrics. We have found that the $ls$ criterion is sometimes misleading in the sense that a low value can result even if the functions are not very well aligned. This is the case, for example, in the male growth data under SMR method. Here the $ls = 0.45$, while for our method $ls = 0.64$, even though it is easy to see that latter has performed a better alignment. On the other hand, the $sls$ criterion seems to best correlate with a visual evaluation of the alignment. The neural spike train data is the most challenging and no other method except ours does a good job.

## 6 Summary

In this paper we have described a parameter-free approach for reconstructing underlying signal using given functions with random warpings, scalings, and translations. The basic idea is to use the Fisher-Rao Riemannian metric and the resulting geodesic distance to define a proper distance, called elastic distance, between warping orbits of SRVF functions. This distance is used to compute a Karcher mean of the orbits, and a template is selected from the mean orbit using an additional condition that the mean of the warping functions is identity. By applying these warpings on the original functions, we provide a consistent estimator of the underlying signal. One interesting application of this framework is in aligning functions with significant x-variability. We show the the proposed Fisher-Rao method provides better alignment performance than the state-of-the-art methods in several real experimental data.

# References

[1] S. Amari. *Differential Geometric Methods in Statistics*. Lecture Notes in Statistics, Vol. 28. Springer, 1985.

[2] N. N. Čencov. *Statistical Decision Rules and Optimal Inferences*, volume 53 of *Translations of Mathematical Monographs*. AMS, Providence, USA, 1982.

[3] B. Efron. Defining the curvature of a statistical problem (with applications to second order efficiency). *Ann. Statist.*, 3:1189–1242, 1975.

[4] D. Gervini and T. Gasser. Self-modeling warping functions. *Journal of the Royal Statistical Society, Ser. B*, 66:959–971, 2004.

[5] G. James. Curve alignment by moments. *Annals of Applied Statistics*, 1(2):480–501, 2007.

[6] R. E. Kass and P. W. Vos. *Geometric Foundations of Asymptotic Inference*. John Wiley & Sons, Inc., 1997.

[7] A. Kneip and T. Gasser. Statistical tools to analyze data representing a sample of curves. *The Annals of Statistics*, 20:1266–1305, 1992.

[8] A. Kneip and J. O. Ramsay. Combining registration and fitting for functional models. *Journal of American Statistical Association*, 103(483), 2008.

[9] J. O. Ramsay and X. Li. Curve registration. *Journal of the Royal Statistical Society, Ser. B*, 60:351–363, 1998.

[10] C. R. Rao. Information and accuracy attainable in the estimation of statistical parameters. *Bulletin of Calcutta Mathematical Society*, 37:81–91, 1945.

[11] R. Tang and H. G. Muller. Pairwise curve synchronization for functional data. *Biometrika*, 95(4):875–889, 2008.

[12] H.L. Van Trees. *Detection, Estimation, and Modulation Theory, vol. I*. John Wiley, N.Y., 1971.

[13] M. Tsang, J. H. Shapiro, and S. Lloyd. Quantum theory of optical temporal phase and instantaneous frequency. *Phys. Rev. A*, 78(5):053820, Nov 2008.

[14] K. Wang and T. Gasser. Alignment of curves by dynamic time warping. *Annals of Statistics*, 25(3):1251–1276, 1997.

[15] A. Willsky. Fourier series and estimation on the circle with applications to synchronous communication–I: Analysis. *IEEE Transactions on Information Theory*, 20(5):577 – 583, sep 1974.

[16] W. Wu and A. Srivastava. Towards Statistical Summaries of Spike Train Data. *Journal of Neuroscience Methods*, 195:107–110, 2011.

